# Classification in Non-Metric Spaces

**Daphna Weinshall**[1,2]  **David W. Jacobs**[1]  **Yoram Gdalyahu**[2]
[1]NEC Research Institute, 4 Independence Way, Princeton, NJ 08540, USA
[2]Inst. of Computer Science, Hebrew University of Jerusalem, Jerusalem 91904, Israel

## Abstract

A key question in vision is how to represent our knowledge of previously encountered objects to classify new ones. The answer depends on how we determine the similarity of two objects. Similarity tells us how relevant each previously seen object is in determining the category to which a new object belongs. Here a dichotomy emerges. Complex notions of similarity appear necessary for cognitive models and applications, while simple notions of similarity form a tractable basis for current computational approaches to classification. We explore the nature of this dichotomy and why it calls for new approaches to well-studied problems in learning. We begin this process by demonstrating new computational methods for supervised learning that can handle complex notions of similarity. (1) We discuss how to implement parametric methods that represent a class by its *mean* when using non-metric similarity functions; and (2) We review non-parametric methods that we have developed using nearest neighbor classification in non-metric spaces. Point (2), and some of the background of our work have been described in more detail in [8].

## 1 Supervised Learning and Non-Metric Distances

*How can one represent one's knowledge of previously encountered objects in order to classify new objects?* We study this question within the framework of supervised learning: it is assumed that one is given a number of *training* objects, each labeled as belonging to a category; one wishes to use this experience to label new *test* instances of objects. This problem emerges both in the modeling of cognitive processes and in many practical applications. For example, one might want to identify risky applicants for credit based on past experience with clients who have proven to be good or bad credit risks. Our work is motivated by computer vision applications.

Most current computational approaches to supervised learning suppose that objects can be thought of as vectors of numbers, or equivalently as points lying in an $n$-dimensional space. They further suppose that the similarity between objects can be determined from the Euclidean distance between these vectors, or from some other simple metric. This classic notion of similarity as Euclidean or metric distance leads

to considerable mathematical and computational simplification.

However, work in cognitive psychology has challenged such simple notions of similarity as models of human judgment, while applications frequently employ non-Euclidean distances to measure object similarity. We consider the need for similarity measures that are not only non-Euclidean, but that are non-*metric*. We focus on proposed similarities that violate one requirement of a metric distance, the triangle inequality. This states that if we denote the distance between objects $A$ and $B$ by $d(A, B)$, then: $\forall A, B, C : d(A, B) + d(B, C) \geq d(A, C)$. Distances violating the triangle inequality must also be non-Euclidean.

Data from cognitive psychology has demonstrated that similarity judgments may not be well modeled by Euclidean distances. Tversky [12] has demonstrated instances in which similarity judgments may violate the triangle inequality. For example, close similarity between Jamaica and Cuba and between Cuba and Russia does not imply close similarity between Jamaica and Russia (see also [10]). Non-metric similarity measures are frequently employed for practical reasons, too (cf. [5]). In part, work in robust statistics [7] has shown that methods that will survive the presence of outliers, which are extraneous pieces of information or information containing extreme errors, must employ non-Euclidean distances that in fact violate the triangle inequality; related insights have spurred the widespread use of robust methods in computer vision (reviewed in [5] and [9]).

We are interested in handling a wide range of non-metric distance functions, including those that are so complex that they must be treated as a black box. However, to be concrete, we will focus here on two simple examples of such distances:

**median distance:** This distance assumes that objects are representable as a set of features whose individual differences can be measured, so that the difference between two objects is representable as a vector: $\vec{d} = (d_1, d_2, ...d_n)$. The median distance between the two objects is just the median value in this vector. Similarly, one can define a *k-median* distance by choosing the $k$'th lowest element in this list. *k*-median distances are often used in applications (cf. [9]), because they are unaffected by the exact values of the most extreme differences between the objects. Only these features that are most similar determine its value. The $k$-median distance can violate the triangle inequality to an arbitrary degree (i.e., there are no constraints on the pairwise distances between three points).

**robust non-metric $L^p$ distances:** Given a difference vector $\vec{d}$, an $L^p$ distance has the form:

$$\left( \sum_{i=1}^{n} d_i^p \right)^{\frac{1}{p}} \tag{1}$$

and is non-metric for $p < 1$.

Figure 1 illustrates why these distances present significant new challenges in supervised learning. Suppose that given some datapoints (two in Fig. 1), we wish to classify each new point as coming from the same category as its nearest neighbor. Then we need to determine the Voronoi diagram generated by our data: a division of the plane into regions in which the points all have the same nearest neighbor. Fig. 1 shows how the Voronoi diagram changes with the function used to compute the distance between datapoints; the non-metric diagrams (rightmost three pictures in Fig. 1) are more complex and more likely to make non-intuitive predictions. In fact, very little is known about the computation of non-metric Voronoi diagrams.

We now describe new parametric methods for supervised learning with non-metric

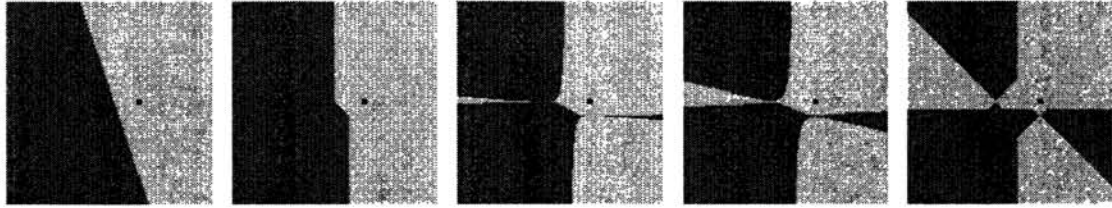

Figure 1: The Voronoi diagram for two points using, from left to right, p-distances with $p = 2$ (Euclidean), $p = 1$ ( Manhattan, which is still metric), the non-metric distances arising from $p = 0.5$, $p = 0.2$, and the min (1-median) distance. The min distance in 2-D illustrates the behavior of the other median distances in higher dimensions. The region of the plane closer to one point is shown in black, and closer to the other in white.

distances, and review non-parametric methods that we described in [8].

## 2  Parametric methods: what should replace the mean

Parametric methods typically represent objects as vectors in a high-dimensional space, and represent classes and the boundaries between them in this space using geometric constructions or probability distributions with a limited number of parameters. One can attempt to extend these techniques to specific non-metric distances, such as the median distance, or non-metric $L^p$ distances. We discuss the example of the mean of a class below. One can also redefine geometric objects such as linear separators, for specific non-metric distances. However, existing algorithms for finding such objects in Euclidean spaces will no longer be directly suitable, nor will theoretical results about such representations hold. Many problems are therefore open in determining how to best apply parametric supervised learning techniques to specific non-metric distances.

We analyze k-means clustering where each class is represented by its average member; new elements are then classified according to which of these prototypical examples is nearest. In Euclidean space, the mean is the point $\bar{q}$ whose sum of squared distances to all the class members $\{q_i\}_{i=1}^n$ - $\left(\sum_{i=1}^n d(\bar{q}, q_i)^2\right)^{\frac{1}{2}}$ - is minimized.

Suppose now that our data come from a vector space where the correct distance is the $L^p$ distance from (1). Using the natural extension of the above definition, we should represent each class by the point $\bar{q}$ whose sum of distances to all the class members - $\left(\sum_{i=1}^n d(\bar{q}, q_i)^p\right)^{\frac{1}{p}}$ - is minimal. It is now possible to show (proof is omitted) that for $p < 1$ (the non-metric cases), the exact value of every feature of the representative point $\bar{q}$ must have already appeared in at least one element in the class. Moreover, the value of these features can be determined separately with complexity $O(n^2)$, and total complexity of $O(dn^2)$ given $d$ features. $\bar{q}$ is therefore determined by a mixture of up to $d$ exemplars, where $d$ is the dimension of the vector space. Thus there are efficient algorithms for finding the "mean" element of a class, even using certain non-metric distances.

We will illustrate these results with a concrete example using the corel database, a commercial database of images pre-labeled by categories (such as "lions"), where non-metric distance functions have proven effective in determining the similarity of images [1]. The corel database is very large, making the use of prototypes desirable.

We represent each image using a vector of 11 numbers describing general image properties, such as color histograms, as described in [1]. We consider the Euclidean

and $L^{0.5}$ distances, and their corresponding prototypes: the mean and the $L^{0.5}$-prototype computed according to the result above. Given the first 45 classes, each containing 100 images, we found their corresponding prototypes; we then computed the percentage of images in each class that are closest to their own prototype, using either the Euclidean or the $L^{0.5}$ distance and one of the two prototypes. The results are the following:

| prototype: | mean | $d$ existing features |
|---|---|---|
| $L^{0.5}$ distance | 18% | 25% |
| Euclidean distance | 20% | 20% |

In the first column, the prototype is computed using the Euclidean mean. In the second column the prototype is computed using an $L^{0.5}$ distance. In each row, a different function is used to compute the distance from each item to the cluster prototype. Best results are indeed obtained with the non-metric $L^{0.5}$ distance and the correct prototype for this particular distance. While performance in absolute terms depends on how well this data clusters using distances derived from a simple feature vector, relative performance of different methods reveals the advantage of using a prototype computed with a non-metric distance.

Another important distance function is the generalized Hamming distance: given two vectors of features, their distance is the number of features which are different in the two vectors. This distance was assumed in psychophysical experiments which used artificial objects (Fribbles) to investigate human categorization and object recognition [13]. In agreement with experimental results, the prototype $\bar{q}$ for this distance computed according to the definition above is the vector of "modal" features - the most common feature value computed independently at each feature.

## 3 Non-Parametric Methods: Nearest Neighbors

Non-parametric classification methods typically represent a class directly by its exemplars. Specifically, nearest-neighbor techniques classify new objects using only their distance to labeled exemplars. Such methods can be applied using any non-metric distance function, treating the function as a black-box. However, nearest-neighbor techniques must also be modified to apply well to non-metric distances. The insights we gain below from doing this can form the basis of more efficient and effective computer algorithms, and of cognitive models for which examples of a class are worth remembering. This section summarizes work described in [8].

Current efficient algorithms for finding the nearest neighbor of a class work only for metric distances [3]. The alternative of a brute-force approach, in which a new object is explicitly compared to every previously seen object, is desirable neither computationally nor as a cognitive model. A natural approach to handling this problem is to represent each class by a subset of its labeled examples. Such methods are called *condensing* algorithms. Below we develop condensing methods for selecting a subset of the training set which minimizes errors in the classification of new datapoints, taking into account the non-metric nature of the distance.

In designing a condensing method, one needs to answer the question *when is one object a good substitute for another?* Earlier methods (e.g., [6, 2]) make use of the fact that the triangle inequality guarantees that when two points are similar to each other, their pattern of similarities to other points are not very different. Thus, in a metric space, there is no reason to store two similar datapoints, one can easily substitute for the other. Things are different in non-metric spaces.

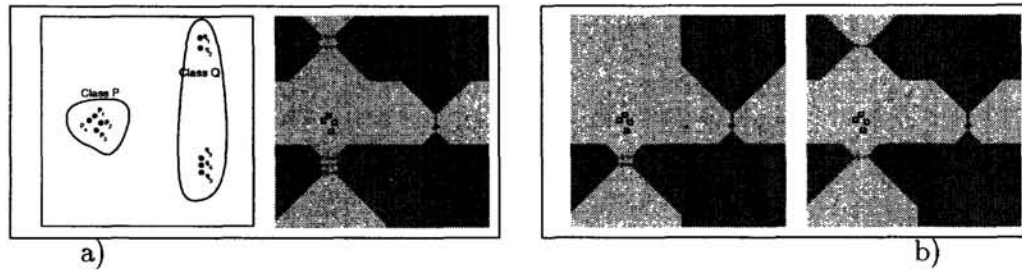

Figure 2: a) Two clusters of labeled points (left) and their Voronoi diagram (right) computed using the 1-median (min) distance. Cluster $P$ consists of four points (black squares) all close together both according to the median distance and the Euclidean distance. Cluster $Q$ consists of five points (black crosses) all having the same $x$ coordinate, and so all are separated by zero distance using the median (but not Euclidean) distance. We wish to select a subset of points to represent each class, while changing this Voronoi diagram as little as possible. b) All points in class $Q$ have zero distance to each other, using the min distance. So distance provides no clue as to which are interchangeable. However, the top points $(q_1, q_2)$ have distances to the points in class $P$ that are highly correlated with each other, and poorly correlated with the bottom points $(q_3, q_4, q_5)$. Without using correlation as a clue, we might represent $Q$ with two points from the bottom (which are nearer the boundary with $P$, a factor preferred in existing approaches). This changes the Voronoi diagram drastically, as shown on the left. Using correlation as a clue, we select points from the top and bottom, changing the Voronoi diagram much less, as shown on the right.

Specifically, what we really need to know is when two objects will have similar distances to other objects, yet unseen. We estimate this quantity using the correlation between two vectors: the vector of distances from one datapoint to all the other training data, and the vector of distances from the second datapoint to all the remaining training data[1]. It can be shown (proof is omitted) that in a Euclidean space the similarity between two points is the best measure of how well one can substitute the other, whereas in a non-metric space the aforementioned vector correlation is a substantially better measure. Fig. 2 illustrates this result.

We now draw on these insights to produce concrete methods for representing classes in non-metric spaces, for nearest neighbor classification. We compare three algorithms. The first two algorithms, **random selection** (cf. [6]) and **boundary detection** (e.g., [11]), represent old condensing ideas: in the first we pick a random selection of class representatives, in the second we use points close to class boundaries as representatives. The last algorithm uses new ideas: **correlation selection** includes in the representative set points which are least correlated with the other class members and representatives. To be fair in our comparison, all algorithms were constrained to select the same number of representative points for each class.

During the simulation, each of 1000 test datapoints was classified based on: (1) all the data, (2) the representatives computed by each of the three algorithms. For each algorithm, the test is successful if the two methods (classification based on all the data and based on the chosen representatives) give the same results. Fig. 3a-c summarizes representative results of our simulations. See [8] for details.

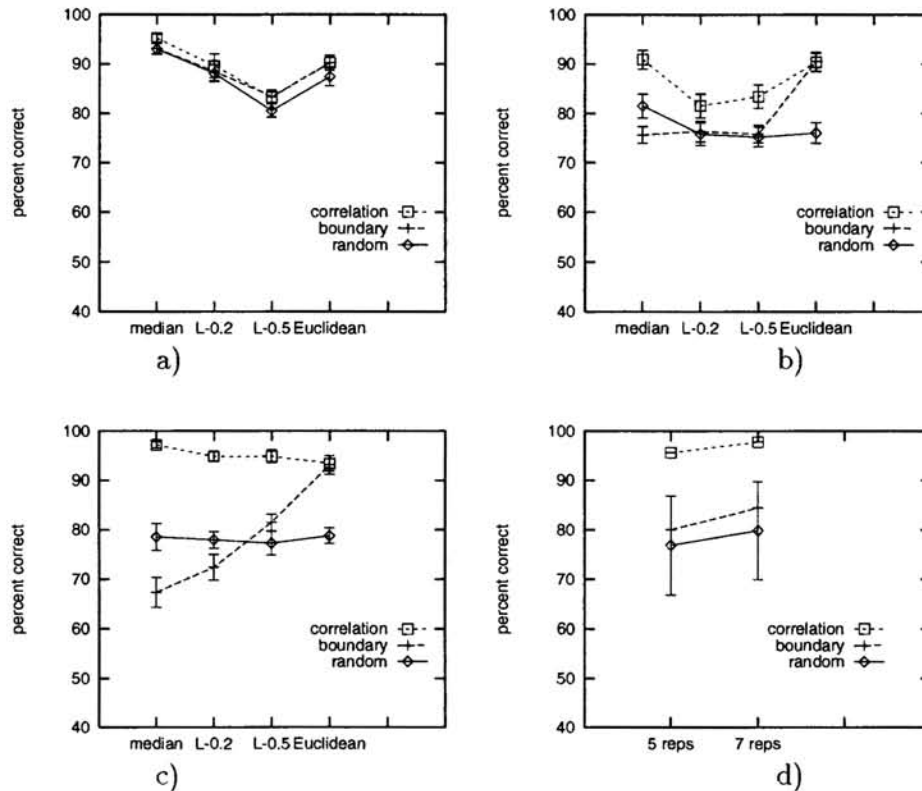

Figure 3: Results: values of percent correct scores, as well as error bars giving the standard deviation calculated over 20 repetitions of each test block when appropriate. Each graph contains 3 plots, giving the percent correct score for each of the three algorithms described above: random (selection), boundary (detection), and (selection based on) correlation. (a-c) Simulation results: data is chosen from $\mathcal{R}^{25}$. 30 clusters were randomly chosen, each with 30 datapoints. The distribution of points in each class was: (a) normal; (b) normal, where in half the datapoints one random coordinate was modified (thus the points cluster around a prototype, but many class members vary widely in one random dimension); (c) union of 2 concentric normal distributions, one spherical and one elongated elliptical (thus the points cluster around a prototype, but may vary significantly in a few non-defining dimensions). Each plot gives 4 values, for each of the different distance functions used here: median, $L^{0.2}$, $L^{0.5}$ and $L^2$. (d) Real data: the number of representatives chosen by the algorithm was limited to 5 (first column) and 7 (second column).

To test our method with real images, we used the local curve matching algorithm described in [4]. This non-metric curve matching algorithm was specifically designed to compare curves which may be quite different, and return the distance between them. The training and test data are shown in Fig. 4. Results are given in Fig. 3d.

The simulations and the real data demonstrate a significant advantage to our new method. Almost as important, in metric spaces (4th column in Fig. 3a-c) or when the classes lack any "interesting" structure (Fig. 3a), our method is not worse than existing methods. Thus it should be used to guarantee good performance when the nature of the data and the distance function is not known a priori.

## Footnotes

[1]Given two datapoints $X, Y$ and $\mathbf{x}, \mathbf{y} \in \mathcal{R}^n$, where $\mathbf{x}$ is the vector of distances from $X$ to all the other training points and $\mathbf{y}$ is the corresponding vector for $Y$, we measure the correlation between the datapoints using the statistical correlation coefficient between $\mathbf{x}, \mathbf{y}$: $corr(X, Y) = corr(\mathbf{x}, \mathbf{y}) = \frac{\mathbf{x} - \mu_x}{\sigma_x} \cdot \frac{\mathbf{y} - \mu_y}{\sigma_y}$, where $\mu_x, \mu_y$ denote the mean of $\mathbf{x}, \mathbf{y}$ respectively, and $\sigma_x, \sigma_y$ denote the standard deviation of $\mathbf{x}, \mathbf{y}$ respectively.

# References

[1] Cox, I., Miller, M., Omohundro, S., and Yianilos, P., 1996, "PicHunter: Bayesian Relevance Feedback for Image Retrieval," *Proc. of ICPR*, C:361–369.

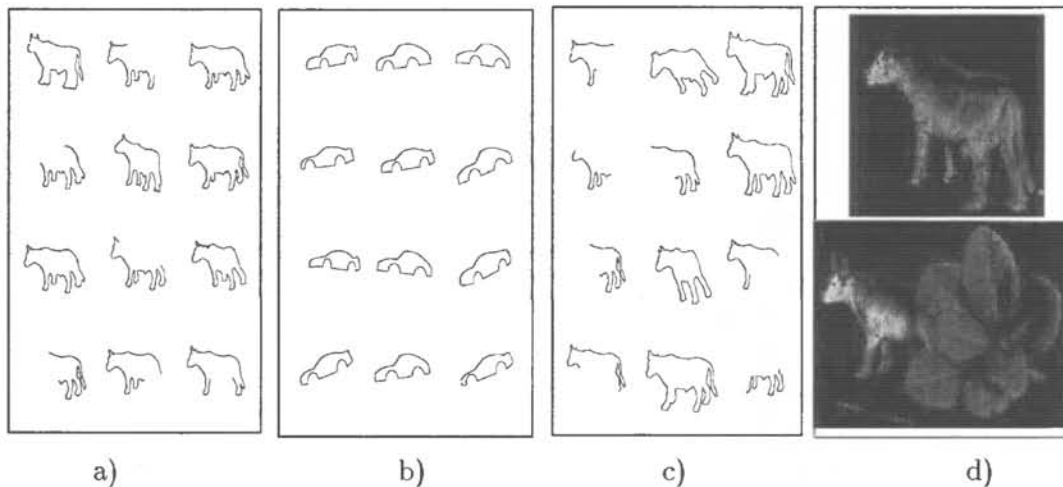

Figure 4: Real data used to test the three algorithms, including 2 classes with 30 images each: a) 12 examples from the first class of 30 cow contours, obtained from different viewpoints of the same cow. b) 12 examples from the second class of 30 car contours, obtained from different viewpoints of 2 similar cars. c) 12 examples from the set of 30 test cow contours, obtained from different viewpoints of the same cow with possibly additional occlusion. d) 2 examples of the real images from which the contours in a) are obtained.

[2] Dasarathy, B., 1994, "Minimal Consistent Set (MCS) Identification for Optimal Near-est Neighbor Decision Systems Design," *IEEE Trans. on Systems, Man and Cyber-netics*, **24**(3):511–517.

[3] Friedman, J., Bently, J., Finkel, R., 1977, "An Algorithm for Finding Best Matches in Logarithmic Expected Time," *ACM Trans. on Math. Software*, **3:3** 209–226.

[4] Gdalyahu, Y. and D. Weinshall, 1997, "Local Curve Matching for Object Recognition without Prior Knowledge", *Proc.: DARPA Image Understanding Workshop*, 1997.

[5] Haralick, R. and L. Shapiro, 1993, *Computer and Robot Vision, Vol. 2*, Addison-Wesley Publishing.

[6] Hart, P., 1968, "The Condensed Nearest Neighbor Rule," *IEEE Trans. on Information Theory*, **14**(3):515–516.

[7] Huber, P., 1981, *Robust Statistics*, John Wiley and Sons.

[8] Jacobs, D., Weinshall, D., and Gdalyahu, Y., 1998, "Condensing Image Databases when Retrieval is based on Non-Metric Distances," *Int. Conf. on Computer vis.*:596–601.

[9] Meer, P., D. Mintz, D. Kim and A. Rosenfeld, 1991, "Robust Regression Methods for Computer Vision: A Review," *Int. J. of Comp. Vis.* **6**(1):59-70.

[10] Rosch, E., 1975, "Cognitive Reference Points," *Cognitive Psychology*, **7**:532–547.

[11] Tomek, I., 1976, "Two modifications of CNN," *IEEE Trans. Syst., Man, Cyber.,*, **SMC-6**(11):769–772.

[12] Tversky, A., 1977, "Features of Similarity," *Psychological Review*, **84**(4):327–352.

[13] Williams, P., "Prototypes, Exemplars, and Object Recognition", submitted.
